# Efficient Neural Codes under Metabolic Constraints

**Zhuo Wang** *[†]
Department of Mathematics
University of Pennsylvania
wangzhuo@nyu.edu

**Xue-Xin Wei** *[‡]
Department of Psychology
University of Pennsylvania
weixxpku@gmail.com

**Alan A. Stocker**
Department of Psychology
University of Pennsylvania
astocker@sas.upenn.edu

**Daniel D. Lee**
Department of Electrical and System Engineering
University of Pennsylvania
ddlee@seas.upenn.edu

## Abstract

Neural codes are inevitably shaped by various kinds of biological constraints, *e.g.* noise and metabolic cost. Here we formulate a coding framework which explicitly deals with noise and the metabolic costs associated with the neural representation of information, and analytically derive the optimal neural code for monotonic response functions and arbitrary stimulus distributions. For a single neuron, the theory predicts a family of optimal response functions depending on the metabolic budget and noise characteristics. Interestingly, the well-known histogram equalization solution can be viewed as a special case when metabolic resources are unlimited. For a pair of neurons, our theory suggests that under more severe metabolic constraints, ON-OFF coding is an increasingly more efficient coding scheme compared to ON-ON or OFF-OFF. The advantage could be as large as one-fold, substantially larger than the previous estimation. Some of these predictions could be generalized to the case of large neural populations. In particular, these analytical results may provide a theoretical basis for the predominant segregation into ON- and OFF-cells in early visual processing areas. Overall, we provide a unified framework for optimal neural codes with monotonic tuning curves in the brain, and makes predictions that can be directly tested with physiology experiments.

## 1 Introduction

The efficient coding hypothesis [1, 2] plays a fundamental role in understanding neural codes, particularly in early sensory processing. Going beyond the original idea of redundancy reduction by Horace Barlow [2], efficient coding has become a general conceptual framework for studying optimal neural coding [3, 4, 5, 6, 7, 8, 9, 10, 11, 12, 13, 14]. Efficient coding theory hypothesizes that the neural code is organized in a way such that maximal information is conveyed about the stimulus variable. Notably, any formulation of efficient coding necessarily relies on a set of constraints due to real world limitations imposed on neural systems. For example, neural noise, metabolic energy budgets, tuning curve characteristics and the size of the neural population all can have impacts on the quality of the neural code.

Most previous studies have only considered a small subset of these constraints. For example, the original redundancy reduction argument proposed by Barlow has focused on utilizing the dynamical

---

[†]current affiliation: Center for Neural Science, New York University
[‡]current affiliation: Department of Statistics and Center for Theoretical Neuroscience, Columbia University

range of the neurons efficiently [2, 15], but did not take neural noise model and energy consumption into consideration. Some studies explicitly dealt with the metabolic costs of the system but did not consider the constraints imposed by the limited firing rates of neurons as well as their detailed tuning properties [16, 7, 17, 18]. As another prominent example, histogram equalization has been proposed as the mechanism for determining the optimal tuning curve of a single neuron with monotonic response characteristics [19]. However, this result only holds for a specific neural noise model and does not take metabolic costs into consideration either. In terms of neural population, most previous studies have focused on bell-shaped tuning curves. Optimal neural coding for neural population with monotonic tuning curves have received much less attention [20, 21].

We develop a formulation of efficient coding that explicitly deals with multiple biologically relevant constraints, including neural noise, limited range of the neural output, and metabolic consumption. With this formulation, we can study neural codes based on monotonic response characteristics that have been frequently observed in biological neural systems. We are able to derive analytical solutions for a wide range of conditions in the small noise limit. We present results for neural populations of different sizes, including the cases of a single neuron, pairs of neurons, as well as a brief treatment for larger neural populations. The results are in general agreements with observed coding schemes for monotonic tuning curves. The results also provide various quantitative predictions which are readily testable with targeted physiology experiments.

## 2 Optimal Code for a Single Neuron

### 2.1 Models and Methods

We start with the simple case where a scalar stimulus $s$ with prior $p(s)$ is encoded by a single neuron. To model the neural response for a stimulus $s$, we first denote the mean output level as a deterministic function $h(s)$. Here $h(s)$ could denote the mean firing rate in the context of rate coding or just the mean membrane potential. In either case, the actual response $r$ is noisy and can be modeled by a probabilistic model $P(r|h(s))$. Throughout the paper, we limit the neural codes to be monotonic functions $h(s)$. The mutual information between the input stimulus $r$ and the neural response is denoted as $\mathrm{MI}(s, r)$.

We formulate the efficient coding problem as the maximization of the mutual information between the stimulus and the response, *e.g.*, $\mathrm{MI}(s, r)$ [3]. To complete the formulation of this problem, it is crucial to choose a set of constraints which characterizes the limited resource available to the neural system. One important constraint is the finite range of the neural output [19]. Another constraint is on the mean metabolic cost [16, 7, 17, 18], which limits the mean activity level of neural output, averaged over the stimulus prior. Under these constraints, the efficient coding problem can mathematically be formulated as following:

$$
\begin{aligned}
\text{maximize} \quad & \mathrm{MI}(s, r) \\
\text{subject to} \quad & 0 \le h(s) \le r_{\max}, \quad h'(s) \ge 0 \qquad \text{(range constraint)} \\
& \mathbf{E}_s[K(h(s))] \le K_{\text{total}} \qquad \text{(metabolic constraint)}
\end{aligned}
$$

We seek the optimal response function $h(s)$ under various choices of the neural noise model $P(r|h(s))$ and certain metabolic cost function $K(h(s))$, as discussed below.

**Neural Noise Models:** Neural noise can often be well characterized by a Poisson distribution at relatively short time scale [22]. Under the Poisson noise model, the number of spikes $N_T$ over a duration of $T$ is a Poisson random variable with mean $h(s)T$ and variance $h(s)T$. In the long $T$ limit, the mean response $r = N_T/T$ approximately follows a Gaussian distribution

$$
r \sim \mathcal{N}(h(s), h(s)/T) \tag{1}
$$

Non-Poisson noise have also been observed physiologically. In these cases, the variance of response $N_T$ can be greater or smaller than the mean firing rate [22, 23, 24, 25]. We thus consider a more generic family of noise models parametrized by $\alpha$

$$
r \sim \mathcal{N}(h(s), h(s)^\alpha/T) \tag{2}
$$

This generalized family of noise model naturally includes the additive Gaussian noise case (when $\alpha = 0$), which is useful for describing the stochasticity of the membrane potential of a neuron.

**Metabolic Cost:** We model the metabolic cost $K$ is a power-law function of the neural output

$$K(h(s)) = h(s)^\beta \tag{3}$$

where $\beta > 0$ is a parameter to model how does the energy cost scale up as the neural output is increasing. For a single neuron we will demonstrate with the general energy cost function but when we generalize to the case of multiple neurons, we will assume $\beta = 1$ for simplicity. Note that it does not require extra effort to solve the problem if the cost function takes the general form of $\tilde{K}(h(s)) = K_0 + K_1 h(s)^\beta$, as reported in [26]. This is because of the linear nature of the expectation term in the metabolic constraint.

## 2.2 Derivation of the Optimal $h(s)$

This efficient coding problem can be greatly simplified due to the fact that it is invariant under any re-parameterization of the stimulus variable $s$. We take this advantage by mapping $s$ to another uniform random variable $u \in [0, 1]$ via the cumulative distribution function $u = F(s)$ [27]. If we choose $g(u) = g(F(s)) = h(s)$, it suffices to solve the following new problem which optimizes $g(u)$ for a re-parameterized input $u$ with uniform prior

$$\begin{aligned} \text{maximize} \quad & \text{MI}(u, r) \\ \text{subject to} \quad & 0 \le g(u) \le r_{\max}, \quad g'(u) \ge 0 \\ & \mathbf{E}_u[K(g(u))] \le K_{\text{total}} \end{aligned}$$

Once the optimal form of $g_*(u)$ is obtained, the optimal $h_*(s)$ is naturally given by $g_*(F(s))$. To solve this simplified problem, first we express the objective function in terms of $g(u)$. In the small noise limit (large integration time $T$), the *Fisher information* $I_F(u)$ of the neuron with noise model in Eq. (2) is calculated and the mutual information can be approximated as (see [28, 14])

$$I_F(u) = T\frac{g'(u)^2}{g(u)^\alpha} + O(1) \tag{4}$$

$$\text{MI}(u, r) = H(U) + \frac{1}{2} \int p(u) \log I_F(u)\, du = \frac{1}{2} \int_0^1 \log \frac{g'(u)^2}{g(u)^\alpha}\, du + \frac{1}{2} \log T + O(1/T) \tag{5}$$

where $H(U) = 0$ is the entropy and $p(u) = 1_{\{0 \le u \le 1\}}$ is the density of the uniform distribution. Furthermore, each constraints can be rewritten as integrals of $g'(u)$ and $g(u)$ respectively:

$$g(1) - g(0) = \int_0^1 g'(u)\, du \le r_{\max} \tag{6}$$

$$\mathbf{E}_u[K(g(u))] = \int_0^1 g(u)^\beta\, du \le K_{\text{total}} \tag{7}$$

This form of the problem (Eq. 5-7) can be analytically solved by using the Lagrangian multiplier method and the optimal response function must take the form

$$g(u) = r_{\max} \cdot \left[\frac{1}{a}\gamma_q^{-1}\left(u\gamma_q(a)\right)\right]^{1/\beta}, \quad h(s) = g(F(s)) \tag{8}$$

$$\text{where } q \overset{\text{def}}{=} (1 - \alpha/2)/\beta, \quad \gamma_q(x) \overset{\text{def}}{=} \int_0^x z^{q-1} \exp(-z)\, dz. \tag{9}$$

The function $\gamma_q(x)$ is called the incomplete gamma function and $\gamma_q^{-1}$ is its inverse. Due to space limitation we only present a sketch derivation. Readers who are interested in the detailed proof are referred to the supplementary materials.

Let us now turn to some intuitive conclusions behind this solution (also see Fig.1, in which we have assumed $r_{\max} = 1$ for simplicity). From Eq. (8), it is clear that the optimal solution $g(u)$ depend on the constant $a$ which should be determined by equalizing the metabolic constraint (see the horizontal dash lines in Fig.1a). Furthermore, the optimal solution $h(s)$ depends on the specific input distribution $p(s)$. Depending on the relative magnitude of $r_{\max}$ and $K_{\text{total}}$:

- **Range constraint dominates:** This is the case when there is more than sufficient energy to achieve the optimal solution so that the metabolic constraint becomes completely redundant. Determined by $\alpha, \beta$ and $r_{\max}$, $K_{\text{thre}}$ is the energy consumption of the optimal code with unconstrained metabolic budget. When the available metabolic cost exceeds this threshold $K_{\text{total}} \geq K_{\text{thre}}$, the constant $a$ is very close to zero and the optimal $g(u)$ is proportional to a power function $g(u) = r_{\max} \cdot u^{1/q}$. See red curves in Fig.1.

- **Both constraints:** This is the general case when $K_{\text{total}} \lesssim K_{\text{thre}}$. The constant $a$ is set to the minimum value for which the metabolic constraint is satisfied. See purple curves in Fig.1.

- **Metabolic constraint dominates:** This happens when $K_{\text{total}} \ll K_{\text{thre}}$. In this case $a$ is often very large. See blue curves in Fig.1.

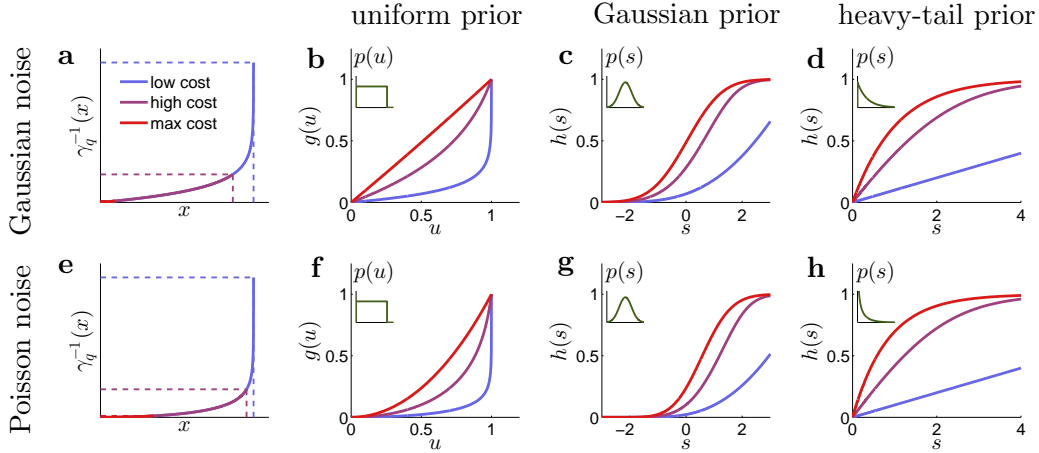

Figure 1: Deriving optimal tuning curves $g(u)$ and corresponding $h(s)$ for different prior distributions and different noise models. Top row: constant Gaussian noise $(\alpha, \beta, q) = (0, 1, 1)$; Bottom row: Poisson noise $(\alpha, \beta, q) = (1, 1, 1/2)$. (a) A segment of the inverse incomplete gamma function is cropped out by dashed boxes. The higher the horizontal dash lines (constant $a$), the lower the average metabolic cost, which corresponds to a more substantial metabolic constraint. We thus use "low","high" and "max" to label the energy costs under different metabolic constraints. (b) The optimal solution $g(u)$ for a uniform variable $u$. (c) The corresponding optimal $h(s)$ for Gaussian prior. (d) The corresponding optimal $h(s)$ for Gamma distribution $p(s) \propto s^{q-1} \exp(-s)$. Specifically for this prior, the optimal tuning curve is exactly linear without maximum response constraint. (e-h) Similar to (a-d), but for Poisson noise.

## 2.3 Properties of the Optimal $h(s)$

We have predicted the optimal response function for arbitrary values of $\alpha$ (which corresponds to the noise model) and $\beta$ (which quantifies the metabolic cost model). Here we specifically focus on a few situations with most biological relevance.

We begin with the simple **additive Gaussian noise** model, i.e. $\alpha = 0$. This model could provide a good characterization of the response mapping from the input stimulus to the membrane potential of a neuron [19]. With more than sufficient metabolic supply, the optimal solution falls back to the histogram equalization principle where each response magnitude is utilized to the same extent (red curve in Fig. 1b and Fig.2a). With less metabolic budget, the optimal tuning curve bends downwards to satisfy this constraint and large responses will be penalized, resulting in more density at smaller response magnitude (purple curve in Fig.2a). In the other extreme, when the available metabolic budget $K_{\text{total}}$ is diminishing, the response magnitude converges to the max-entropy distribution under the metabolic constraint $\mathbf{E}[g(u)^\beta] = const$ (blue curve in Fig.2a).

Next we discuss the case of **Poisson spiking neurons**. In the extreme case when the range constraint dominates, the model predicts a square tuning curve for uniform input (red curve in Fig.1f), which is consistent with previous studies [29, 30]. We also found that Poisson noise model leads to heavier

penalization on large response magnitude compared to Gaussian noise, suggesting an interaction between noise and metabolic cost in shaping the optimal neural response distribution. In the other extreme when $K_{\text{total}}$ goes to 0, the response distribution converges to a gamma distribution, with heavy tail (see Fig.2). Our analytical result gives a simple yet quantitative explanation of the emergence of sparse coding [7] from an energy-efficiency perspective.

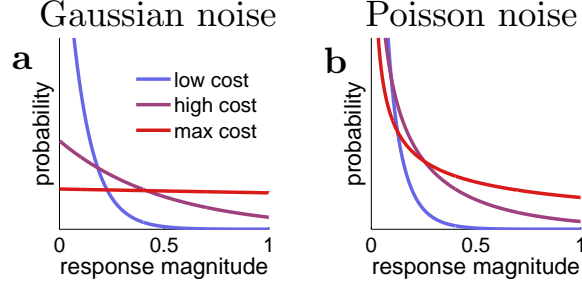

Figure 2: Probability of generating certain response $g(u)$ based on the optimal tuning of a single neuron under (a) Gaussian noise model and (b) Poisson noise model. In the extreme case of Gaussian noise with effectively no metabolic constraint, the distribution is uniformly distributed on the whole range.

## 3 Optimal Code for a Pair of Neurons

We next study the optimal coding in the case of two neurons with monotonic response functions. We denote the neural responses as $\mathbf{r} = (r_1, r_2)$. Therefore the efficient coding problem becomes:

$$\text{maximize} \quad \text{MI}(s, \mathbf{r})$$
$$\text{subject to} \quad 0 \leq h_i(s) \leq r_{\max}, \quad i = 1, 2. \qquad \text{(range constraint)}$$
$$\mathbf{E}_s\left[K(h_1(s)) + K(h_2(s))\right] \leq 2K_{\text{total}} \qquad \text{(metabolic constraint)}$$

Assuming the neural noise is independent across neurons, the system of two neurons has total Fisher information just as the linear sum of Fisher information contributed from each neuron $I_F(s) = I_1(s) + I_2(s)$.

### 3.1 Optimal response functions

Previous studies on neural coding with monotonic response functions have typically assumed that each $h_i(s)$ has sigmoidal shape. It is important to emphasize that we do not make any a priori assumptions on the detailed shape of the tuning curve other than being monotonic and smooth. We define each neuron's active region $A_i = A_i^+ \cup A_i^-$, where $A_i^+ = \{s | h_i'(s) > 0\}$, $A_i^- = \{s | -h_i'(s) > 0\}$. Due to the monotonicity of tuning curve, either $A_i^+$ or $A_i^-$ has to be empty.

We find the following results (proof in the supplementary materials)

1. Different neurons should have non-overlapping active regions.
2. If the metabolic constraint is binding, ON-OFF coding is better than ON-ON coding or OFF-OFF coding. Otherwise all three coding schemes can achieve the same mutual information.
3. For ON-OFF coding, it is better to have ON regions on the right side.
4. For ON-ON coding (or OFF-OFF), each neuron should have roughly the same tuning curve $h_i(s) \approx h_j(s)$ while still have disjoint active regions. Note that a conceptually similar coding scheme has been previously discussed by [29]. Within the ON-pool or OFF-pool, the optimal tuning curve is same as the optimal solution from the single neuron case.

In Fig.3a-d, we illustrate how these conclusions can be used to determine the optimal pair of neurons, assuming additive Gaussian noise $\alpha = 0$ and linear metabolic cost $\beta = 1$ (for other $\alpha$ and $\beta$ the process is similar). Our analytical results allow us to predict the precise shape of the optimal response functions, which goes beyond previous work on ON-OFF coding schemes [13, 31].

## 3.2 Comparison between ON-OFF and ON-ON codes

We aim to compare the coding performance of ON-OFF and ON-ON codes. In Fig.3e we show how the mutual information depends on the available metabolic budget. For both ON-FF and ON-ON scheme, the mutual information is monotonically increasing as a function of energy available. We compare these two curves in two different ways. First, we notice that both mutual information curve saturate the limit at $K_{\text{ON-ON}} = 0.5r_{\text{max}}$ and $K_{\text{ON-OFF}} = 0.25r_{\text{max}}$ respectively (see the red tuning curves in Fig.3a-d). Note that this specific saturation limit is only valid for $\alpha = 0$ and $\beta = 1$. For any other mutual information, we find out that the optimal ON-ON pair (or OFF-OFF pair) always cost twice energy compared to the optimal ON-OFF pair. Second, one can compare the ON-ON and ON-OFF scheme by fixing the energy available. The optimal mutual information achieved by ON-ON neurons is always smaller than that achieved by ON-OFF neurons and the difference is plotted in Fig.3. When the available energy is extremely limited $K_{\text{total}} \ll r_{\text{max}}$, such difference saturates at $-1$ in the logarithm space of MI (base 2). This shows that, in the worst scenario, the ON-ON code is only half as efficient as the ON-OFF code from mutual information perspective. In other words, it would take twice the amount of time $T$ for the ON-ON code to convey same amount of mutual information as the ON-OFF code under same noise level.

These analyses quantitatively characterize the advantage of ON-OFF over ON-ON and show how it varies when the relative importance of the metabolic constraint changes. The encoding efficiency of ON-OFF ranges from double (with very limited metabolic budget) to equal amount of the ON-ON efficiency (with unlimited metabolic budget). This wide range includes the previous conclusion reported by Gjorgjieva *et.al.*, where a mild advantage ($\sim 15\%$) of ON-OFF scheme is found under short integration time limit [31]. It is well known that the split of ON and OFF pathways exists in

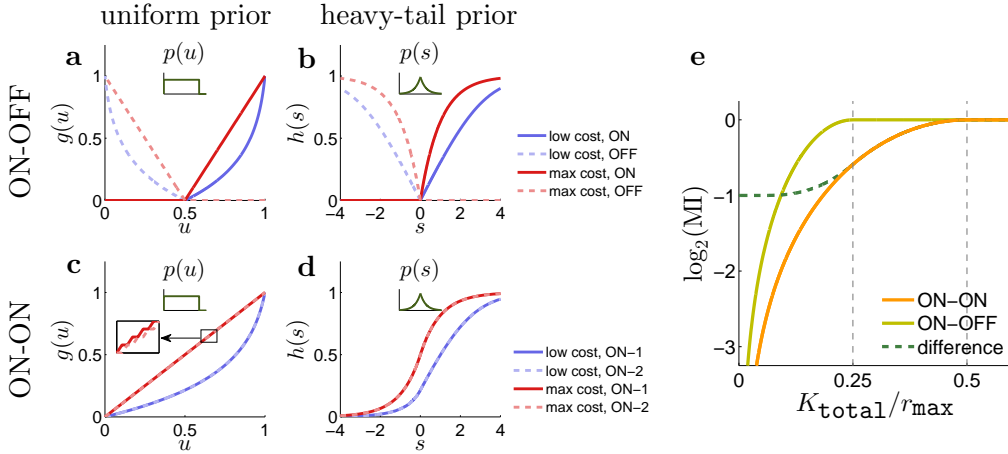

Figure 3: The optimal response functions for a pair of neurons assuming Gaussian noise. (a) The optimal response functions for a uniform input distribution assuming ON-OFF coding scheme. Solid red curve and dash red curve represent the optimal response functions for a pair of neurons with no metabolic constraint ("max cost"). Solid blue and dash blue curves are the optimal response functions with substantial metabolic constraint ("low cost"). (b) Similar to panel a, but for input stimuli with heavy tail distribution. (c) The optimal response functions for a uniform input distribution assuming ON-ON coding scheme. Solid and dash red curves are for no metabolic constraint. Notice that two curves appear to be identical but are actually different at finer scales (see the inserted panel). Solid and dash blue are for substantial metabolic constraint. (d) Similar to panel c, but for input stimuli with heavy tail distribution. (e) A comparison between the ON-ON and ON-OFF schemes. The $x$-axis represents how substantial the metabolic constraint is – any value greater than the threshold 0.5 implies no metabolic constraint in effect. The $y$-axis represents the mutual information, relative to the maximal achievable mutual information without metabolic constraints (which is the same for ON-ON and ON-OFF schemes). The green dash line represents the difference between the information transmitted by the two schemes. Negative difference indicates an advantage of ON-OFF over ON-ON.

the retina of many species [32, 33]. The substantial increase of efficiency under strong metabolic constraint we discovered supports the argument that metabolic constraint may be one of the main reasons for such pathway splitting in evolution.

In a recent study by Karklin and Simoncelli [13], it is observed numerically that ON-OFF coding scheme can naturally emerge when a linear-nonlinear population of neurons are trained to maximize mutual information with image input and under metabolic constraint. It is tempting to speculate a generic connection of these numerical observations to our theoretical results, although our model is much more simplified in the sense that we do not directly model the higher dimensional stimulus (natural image) but just a one dimensional projection (local contrast). Intriguingly, we find that if the inputs follow certain heavy tail distribution ( Fig.3b), the optimal response functions are two rectified non-linear functions which split the encoding range. Such rectified non-linearity is consistent with both the non-linearity observed physiologically[34] and the numerical results in [13] .

## 4   Discussion

In this paper we presented a theoretical framework for studying optimal neural codes under biologically relevant constraints. Compared to previous works, we emphasize the importance of two types of constraint – the noise characteristics of the neural responses and the metabolic cost. Throughout the paper, we have focused on neural codes with smooth monotonic response functions. We demonstrated that, maybe surprisingly, analytical solutions exist for a wide family of noise characteristics and metabolic cost functions. These analytical results rely on the techniques of approximating mutual information using Fisher information. There are cases when such approximation would bread down, in particular for short integration time or non-Gaussian noise. For a more detailed discussion on the validity of Fisher approximation, see [29, 14, 35].

We have focused on the cases of a single neuron and a pair of neurons. However, the framework can be generalized to the case of larger population of neurons. For the case of $N = 2k$ ($k$ is large) neurons, we again find the corresponding optimization problem could be solved analytically by exploiting the Fisher information approximation of mutual information [28, 14]. Interestingly, we found the optimal codes should be divided into two pools of neurons of equal size $k$. One pool of neuron with monotonic increasing response function (ON-pool), and the other with monotonic decreasing response function (OFF-pool). For neurons within the same pool, the optimal response functions appear to be identical on the macro-scale but are quite different when zoomed in. In fast, the optimal code must have disjoint active regions for each neuron. This is similar to what has been illustrated in the inset panel of Fig.3c, where two seemingly identical tuning curves for ON-neurons are compared. We can also quantify the increase of the mutual information by using optimal coding schemes versus using all ON neurons (or all OFF). Interestingly, some of the key results presented in the Fig 3e for the a pair of neurons generalize to $2K$ case. When $N = 2k + 1$, the optimal solution is similar to $N = 2k$ for a large pool of neurons. However, when $k$ is small, the difference caused by asymmetry between ON/OFF pools can substantially change the configuration of the optimal code.

Due to the limited scope of the paper, we have ignored several important aspects when formulating the efficient coding problem. First, we have not modeled the spontaneous activity (baseline firing rate) of neurons. Second, we have not considered the noise correlations between the responses of neurons. Third, we have ignored the noise in the input to the neurons. We think that the first two factors are unlikely to change our main results. However, incorporating the input noise may significantly change the results. In particular, for the cases of multiple neurons, our current results predict that there is no overlap between the active regions of the response functions for ON and OFF neurons. However, it is possible that this prediction does not hold in the presence of the input noise. In that case, it might be beneficial to have some redundancy by making the response functions partially overlap. Including these factors into the framework should facilitate a detailed and quantitative comparison to physiologically measured data in the future. As for the objective function, we have only considered the case of maximizing mutual information; it is interesting to see whether the results can be generalized to other objective functions such as, *e.g.*, minimizing decoding error[36, 37]. Also, our theory is based on a one dimensional input. To fully explain the ON-OFF split in visual pathway, it seems necessary to consider a more complete model with the images as the input. To this end, our current model lacks the spatial component, and it doesn't explain the difference between the number of ON and OFF neurons in retina [38]. Nonetheless, the insight from these analytical results based on the simple model may prove to be useful for a more complete understanding of the functional

organization of the early visual pathway. Last but not least, we have assumed a stationary input distribution. However, in the natural environment the input distribution often fluctuate at different time scales, it remains to be investigated how to incorporate these dynamical aspects into a theory of efficient coding.

# References

[1] Fred Attneave. Some informational aspects of visual perception. *Psychological review*, 61(3):183, 1954.

[2] Horace B Barlow. Possible principles underlying the transformation of sensory messages. *Sensory communication*, pages 217–234, 1961.

[3] Ralph Linsker. Self-organization in a perceptual network. *Computer*, 21(3):105–117, 1988.

[4] Joseph J Atick and A Norman Redlich. Towards a theory of early visual processing. *Neural Computation*, 2(3):308–320, 1990.

[5] Joseph J Atick. Could information theory provide an ecological theory of sensory processing? *Network: Computation in neural systems*, 3(2):213–251, 1992.

[6] F Rieke, DA Bodnar, and W Bialek. Naturalistic stimuli increase the rate and efficiency of information transmission by primary auditory afferents. *Proceedings of the Royal Society of London. Series B: Biological Sciences*, 262(1365):259–265, 1995.

[7] Bruno Olshausen and David Field. Emergence of simple-cell receptive field properties by learning a sparse code for natural images. *Nature*, 381:607–609, 1996.

[8] Anthony J Bell and Terrence J Sejnowski. The "independent components" of natural scenes are edge filters. *Vision research*, 37(23):3327–3338, 1997.

[9] Eero P Simoncelli and Bruno A Olshausen. Natural image statistics and neural representation. *Annual review of neuroscience*, 24(1):1193–1216, 2001.

[10] Allan Gottschalk. Derivation of the visual contrast response function by maximizing information rate. *Neural computation*, 14(3):527–542, 2002.

[11] Nicol S Harper and David McAlpine. Optimal neural population coding of an auditory spatial cue. *Nature*, 430(7000):682–686, 2004.

[12] Mark D McDonnell and Nigel G Stocks. Maximally informative stimuli and tuning curves for sigmoidal rate-coding neurons and populations. *Physical review letters*, 101(5):058103, 2008.

[13] Yan Karklin and Eero P Simoncelli. Efficient coding of natural images with a population of noisy linear-nonlinear neurons. *Advances in neural information processing systems*, 24:999, 2011.

[14] Xue-Xin Wei and Alan A Stocker. Mutual information, fisher information, and efficient coding. *Neural computation*, 2016.

[15] Horace Barlow. Redundancy reduction revisited. *Network: computation in neural systems*, 12(3):241–253, 2001.

[16] William B Levy and Robert A Baxter. Energy efficient neural codes. *Neural computation*, 8(3):531–543, 1996.

[17] Simon B Laughlin, Rob R de Ruyter van Steveninck, and John C Anderson. The metabolic cost of neural information. *Nature neuroscience*, 1(1):36–41, 1998.

[18] Vijay Balasubramanian, Don Kimber, and Michael J Berry II. Metabolically efficient information processing. *Neural Computation*, 13(4):799–815, 2001.

[19] Simon B Laughlin. A simple coding procedure enhances a neuron's information capacity. *Z. Naturforsch*, 36(910-912):51, 1981.

[20] Deep Ganguli and Eero P Simoncelli. Efficient sensory encoding and Bayesian inference with heterogeneous neural populations. *Neural Computation*, 26(10):2103–2134, 2014.

[21] David B Kastner, Stephen A Baccus, and Tatyana O Sharpee. Critical and maximally informative encoding between neural populations in the retina. *Proceedings of the National Academy of Sciences*, 112(8):2533–2538, 2015.

[22] George J Tomko and Donald R Crapper. Neuronal variability: non-stationary responses to identical visual stimuli. *Brain research*, 79(3):405–418, 1974.

[23] DJ Tolhurst, JA Movshon, and ID Thompson. The dependence of response amplitude and variance of cat visual cortical neurones on stimulus contrast. *Experimental brain research*, 41(3-4):414–419, 1981.

[24] Mark M Churchland et al. Stimulus onset quenches neural variability: a widespread cortical phenomenon. *Nature neuroscience*, 13(3):369–378, 2010.

[25] Moshe Gur and D Max Snodderly. High response reliability of neurons in primary visual cortex (v1) of alert, trained monkeys. *Cerebral cortex*, 16(6):888–895, 2006.

[26] David Attwell and Simon B Laughlin. An energy budget for signaling in the grey matter of the brain. *Journal of Cerebral Blood Flow & Metabolism*, 21(10):1133–1145, 2001.

[27] Xue-Xin Wei and Alan A Stocker. A bayesian observer model constrained by efficient coding can explain'anti-bayesian'percepts. *Nature Neuroscience*, 2015.

[28] Nicolas Brunel and Jean-Pierre Nadal. Mutual information, Fisher information, and population coding. *Neural Computation*, 10(7):1731–1757, 1998.

[29] Matthias Bethge, David Rotermund, and Klaus Pawelzik. Optimal short-term population coding: when fisher information fails. *Neural Computation*, 14(10):2317–2351, 2002.

[30] Don H Johnson and Will Ray. Optimal stimulus coding by neural populations using rate codes. *Journal of computational neuroscience*, 16(2):129–138, 2004.

[31] Julijana Gjorgjieva, Haim Sompolinsky, and Markus Meister. Benefits of pathway splitting in sensory coding. *The Journal of Neuroscience*, 34(36):12127–12144, 2014.

[32] Peter H Schiller. The on and off channels of the visual system. *Trends in neurosciences*, 15(3):86–92, 1992.

[33] Heinz Wässle. Parallel processing in the mammalian retina. *Nature Reviews Neuroscience*, 5(10):747–757, 2004.

[34] Matteo Carandini. Amplification of trial-to-trial response variability by neurons in visual cortex. *PLoS Biol*, 2(9):e264, 2004.

[35] Zhuo Wang, Alan A Stocker, and Daniel D Lee. Efficient neural codes that minimize lp reconstruction error. *Neural Computation*, 2016.

[36] Tvd Twer and Donald IA MacLeod. Optimal nonlinear codes for the perception of natural colours. *Network: Computation in Neural Systems*, 12(3):395–407, 2001.

[37] Zhuo Wang, Alan A Stocker, and Daniel D Lee. Optimal neural tuning curves for arbitrary stimulus distributions: Discrimax, infomax and minimum $L_p$ loss. In *Advances in Neural Information Processing Systems NIPS*, pages 2177–2185, 2012.

[38] Charles P Ratliff, Bart G Borghuis, Yen-Hong Kao, Peter Sterling, and Vijay Balasubramanian. Retina is structured to process an excess of darkness in natural scenes. *Proceedings of the National Academy of Sciences*, 107(40):17368–17373, 2010.

